# The Steering Approach for Multi-Criteria Reinforcement Learning

**Shie Mannor and Nahum Shimkin**
Department of Electrical Engineering
Technion, Haifa 32000, Israel
{shie,shimkin}@{tx,ee}.technion.ac.il

## Abstract

We consider the problem of learning to attain multiple goals in a dynamic environment, which is initially unknown. In addition, the environment may contain arbitrarily varying elements related to actions of other agents or to non-stationary moves of Nature. This problem is modelled as a stochastic (Markov) game between the learning agent and an arbitrary player, with a vector-valued reward function. The objective of the learning agent is to have its long-term average reward vector belong to a given target set. We devise an algorithm for achieving this task, which is based on the theory of approachability for stochastic games. This algorithm combines, in an appropriate way, a finite set of standard, scalar-reward learning algorithms. Sufficient conditions are given for the convergence of the learning algorithm to a general target set. The specialization of these results to the single-controller Markov decision problem are discussed as well.

## 1 Introduction

This paper considers an on-line learning problem for Markov decision processes with vector-valued rewards. Each entry of the reward vector represents a scalar reward (or cost) function which is of interest. Focusing on the long-term average reward, we assume that the desired performance is specified through a given target set, to which the average reward vector should eventually belong. Accordingly, the specified goal of the decision maker is to ensure that the average reward vector will converge to the target set. Following terminology from game theory, we refer to such convergence of the reward vector as *approaching* the target set.

A distinctive feature of our problem formulation is the possible incorporation of arbitrarily varying elements of the environment, which may account for the influence of other agents or non-stationary moves of Nature. These are collectively modelled as a second agent, whose actions may affect both the state transition and the obtained rewards. This agent is free to choose its actions according to any control policy, and no prior assumptions are made regarding its policy.

This problem formulation is derived from the so-called theory of approachability that was introduced in [3] in the context of repeated matrix games with vector payoffs. Using a geometric viewpoint, it characterizes the sets in the reward space that a player can guarantee for himself for *any* possible policy of the other player, and provides appropriate policies for approaching these sets. Approachability theory has been extended to stochastic (Markov) games in [14], and the relevant results are briefly reviewed in Section 2. In this paper we add the learning aspect, and consider the problem of learning such approaching policies on-line, using Reinforcement Learning (RL) or similar algorithms.

Approaching policies are generally required to be non-stationary. Their construction relies on a geometric viewpoint, whereby the average reward vector is "steered" in the direction of the target set by the use of direction-dependent (and possibly stationary) control policies. To motivate the steering viewpoint, consider the following one dimensional example of an automatic temperature

controlling agent. The measured property is the temperature which should be in some prescribed range $[\underline{T}, \overline{T}]$, the agent may activate a cooler or a heater at will. An obvious algorithm that achieves the prescribed temperature range is – when the average temperature is higher than $\overline{T}$ choose a "policy" that reduces it, namely activate the cooler; and if the average temperature is lower than $\underline{T}$ use the heater. See Figure 1(a) for an illustration. Note that this algorithm is robust and requires little knowledge about the characteristics of the processes, as would be required by a procedure that tunes the heater or cooler for continuous operation. A learning algorithm needs only determine which element to use at each of the two extreme regions.

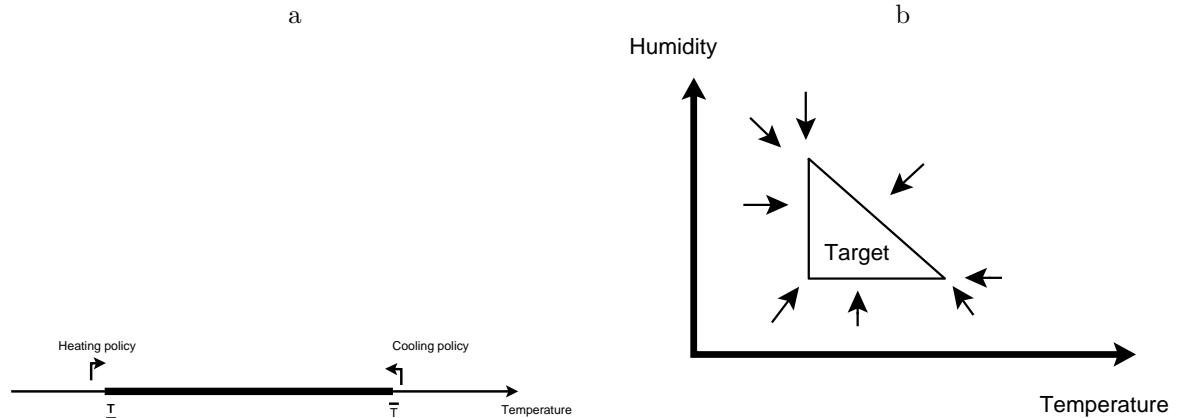

Figure 1: **(a)** The single dimensional temperature example. If the temperature is higher than $\overline{T}$ the control is to cool, and if the temperature is lower than $\underline{T}$ the control is to heat. **(b)** The two dimensional temperature-humidity example. The learning directions are denoted by arrows, note that an infinite number of directions are to be considered.

Consider next a more complex multi-objective version of this controlling agent. The controller's objective is as before to have the temperature in a certain range. One can add other parameters such as the average humidity, frequency of switching between policies, average energy consumption and so on. This problem is naturally characterized as a multi-objective problem, in which the objective of the controller is to have the average reward in some target set. (Note that in this example, the temperature itself is apparently the object of interest rather than its long-term average. However, we can reformulate the temperature requirement as an average reward objective by measuring the fraction of times that the temperature is outside the target range, and require this fraction to be zero. For the purpose of illustration we shall proceed here with the original formulation). For example, suppose that the controller is also interested in the humidity. For the controlled environment of, say, a greenhouse, the allowed level of humidity depends on the average temperature. An illustrative target set is shown in Figure 1(b). A steering policy for the controller is not as simple anymore. In place of the two directions (left/right) of the one-dimensional case, we now face a continuum of possible directions, each associated with a possibly different steering policy. For the purpose of the proposed learning algorithm we shall require to consider only a finite number of steering policies. We will show that this can always be done, with negligible effect on the attainable performance.

The analytical basis for this work relies on three elements: stochastic game models, which capture the Markovian system dynamics while allowing arbitrary variation in some elements of the environment; the theory of approachability for vector-valued dynamic games, which provides the basis for the steering approach; and RL algorithms for (scalar) average reward problems. For the sake of brevity, we do not detail the mathematical models and proofs and concentrate on concepts.

Reinforcement Learning (RL) has emerged in the last decade as a unifying discipline for learning and adaptive control. Comprehensive overviews may be found in [2, 7]. RL for average reward Markov Decision Processes (MDPs) was suggested in [13, 10] and later analyzed in [1]. Several methods exist for average reward RL, including Q-learning [1] the $E^3$ algorithm [8], actor-critic schemes [2] and more.

The paper is organized as follows: In Section 2 we describe the stochastic game setup, recall ap-

proachability theory, and mention a key theorem that allows to consider only a finite number of directions for approaching a set. Section 3 describes the proposed multi-criteria RL algorithm and outlines its convergence proof. We also briefly discuss learning in multi-criteria single controller environments, as this case is a special case of the more general game model. An illustrative example is briefly described in Section 4 and concluding remarks are drawn in Section 5.

## 2 Multi-Criteria Stochastic Games

In this section we present the multi-criteria stochastic game model. We recall some known results from approachability theory for stochastic games with vector-valued reward, and state a key theorem which decomposes the problem of approaching a target set into a finite number of scalar control problems.

We consider a two-person average reward stochastic game model, with a vector-valued reward function. We refer to the players as P1 (the learning agent) and P2 (the arbitrary adversary). The game is defined by: the state space $\mathcal{S}$; the sets of actions for P1 and P2, respectively, in each state $s$, $\mathcal{A}$ and $\mathcal{B}$; the state transition kernel, $P = (P(s'|s,a,b))$; a vector-valued reward function $m : \mathcal{S} \times \mathcal{A} \times \mathcal{B} \to \mathbb{R}^k$. The reward itself is allowed to be random, in which case it is assumed to have a bounded second moment. At each time epoch $n \geq 0$, both players observe the current state $s_n$, and then P1 and P2 simultaneously choose actions $a_n$ and $b_n$, respectively. As a result P1 receives the reward vector $m_n = m(s_n, a_n, b_n)$ and the next state is determined according to the transition probability $P(\cdot|s_n, a_n, b_n)$. More generally, we allow the actual reward $m_n$ to be random, in which case $m(s_n, a_n, b_n)$ denotes its mean and a bounded second moment is assumed. We further assume that both players observe the previous rewards and actions (however, in some of the learning algorithms below, the assumption that P1 observes P2's action may be relaxed). A policy $\pi \in \Pi$ for P1 is a mapping which assigns to each possible observed history a mixed action in $\Delta(\mathcal{A})$, namely a probability vector over P1's action set $\mathcal{A}$. A policy $\sigma \in \Sigma$ for P2 is defined similarly. A policy of either player is called *stationary* if the mixed action it prescribes depends only on the current state $s_n$. Let $\hat{m}_n$ denote the average reward by time $n$: $\hat{m}_n \triangleq \frac{1}{n} \sum_{t=0}^{n-1} m_t$.

The following recurrence assumption will be imposed. Let state $s^*$ denote a specific reference state to which a return is guaranteed. We define the hitting time of state $s^*$ as: $\tau \triangleq \min\{n > 0 : s_n = s^*\}$.

**Assumption 1 (Recurrence)** *There exist a state $s^* \in \mathcal{S}$ and a finite constant $N$ such that*

$$E_{\pi\sigma}^s(\tau^2) < N \quad \text{for all } \pi \in \Pi, \, \sigma \in \Sigma \text{ and } s \in \mathcal{S},$$

*where $E_{\pi\sigma}^s$ is the expectation operator when starting from state $s_0 = s$ and using policies $\pi$ and $\sigma$ for P1 and P2, respectively.*

If the game is finite then this assumption is satisfied if state $s^*$ is accessible from all other states under any pair of stationary deterministic policies [14]. We note that the recurrence assumption may be relaxed in a similar manner to [11].

Let $u$ be a unit vector in the reward space $\mathbb{R}^k$. We often consider the *projected game in direction* $u$ as the zero-sum stochastic game with same dynamic as above, and *scalar* rewards $r_n := m_n \cdot u$. Here "·" stands for the standard inner product in $\mathbb{R}^k$. Denote this game by $\Gamma_s(u)$, where $s$ is the initial state. The scalar stochastic game $\Gamma_s(u)$, has a *value*, denoted $v\Gamma_s(u)$, if

$$v\Gamma_s(u) = \sup_{\pi} \inf_{\sigma} \liminf_{n\to\infty} E_{\pi\sigma}^s(\hat{m}_n \cdot u) = \inf_{\sigma} \sup_{\pi} \limsup_{n\to\infty} E_{\pi\sigma}^s(\hat{m}_n \cdot u).$$

For finite games, the value exists [12]. Furthermore, under Assumption 1 the value is independent of the initial state and can be achieved in stationary policies [6]. We henceforth simply write $v\Gamma(u)$ for this value.

We next consider the task of approaching a given target set in the reward space, and introduce approaching policies for the case where the game parameters are fully known to P1. Let $T \subset \mathbb{R}^k$ denote the target set. In the following, $d$ is the Euclidean distance in $\mathbb{R}^k$, and $P_{\pi,\sigma}^s$ is the probability measure induced by the policies $\pi$ and $\sigma$, with initial state $s$.

**Definition 2.1** *The set $T \subset \mathbb{R}^k$ is approachable (from initial state s) if there exists a T-approaching policy $\pi^*$ of P1 such that $d(\hat{m}_n, T) \to 0 \quad P^s_{\pi^*,\sigma}$-a.s., for every $\sigma \in \Sigma$ at a uniform rate over $\Sigma$.*

The policy $\pi^*$ in that definition will be called an approaching policy for P1. A set is approachable if it is approachable from all states. Noting that approaching a set and its closure are the same, we shall henceforth suppose that the set $T$ is closed.

We recall the basic results from [14] regarding approachability for known stochastic games, which generalize Blackwell's conditions for repeated matrix games. Let

$$\phi(\pi, \sigma) \triangleq \frac{E^{s^*}_{\pi,\sigma}(\sum_{t=0}^{\tau-1} m_t)}{E^{s^*}_{\pi,\sigma}(\tau)} \tag{1}$$

denote the *average per-cycle* reward vector, which is the expected total reward over the cycle that starts and ends in the reference state, divided by the expected duration of that cycle. For any $x \notin T$, denote by $C_x$ a closest point in $T$ to $x$, and let $u_x$ be the unit vector in the direction of $C_x - x$, which points from $x$ to the goal set $T$, see Figure 2 for an illustration.

**Theorem 2.1** [14] *Let Assumption 1 hold. Assume that for every point $x \notin T$ there exists a policy $\pi(x)$ such that:*

$$(\phi(\pi(x), \sigma) - C_x) \cdot u_x \geq 0, \quad \forall \sigma \in \Sigma. \tag{2}$$

*Then $T$ is approachable by P1. An approaching policy is: If $s_n = s^*$ and $\hat{m}_n \notin T$, play $\pi(\hat{m}_n)$ until the next visit to state $s^*$; otherwise, play arbitrarily.*

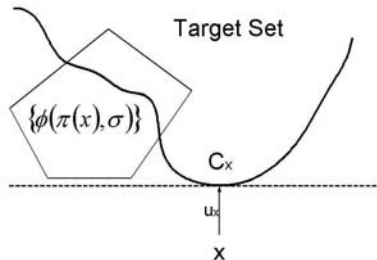

Figure 2: An illustration of approachability. $\pi(x)$ brings P1 to the other side of the hyperplane perpendicular to the segment between $C_x$ and $x$.

Geometrically, the condition in (2) means that P1 can ensure, irrespectively of P2's policy, that the average per-cycle reward will be on the other side (relative to $x$) of the hyperplane which is perpendicular to the line segment that points from $x$ to $C_x$. We shall refer to the direction $u_x$ as the *steering direction* from point $x$, and to the policy $\pi(x)$ as the *steering policy* from $x$. The approaching policy uses the following rule: between successive visits to the reference state, a fixed (possibly stationary) policy is used. When in the reference state, the current average reward vector $\hat{m}_n$ is inspected. If this vector is not in $T$, then the steering policy that satisfies (2) with $x = \hat{m}_n$ is selected for the next cycle. Consequently, the average reward is "steered" towards $T$, and eventually converges to it.

Recalling the definition of the projected game in direction $u$ and its value $v\Gamma(u)$, the condition in (2) may be equivalently stated as $v\Gamma(u_x) \geq C_x \cdot u_x$. Furthermore, the policy $\pi(x)$ can always be chosen as the stationary policy which is optimal for P1 in the game $\Gamma(u_x)$. In particular, the steering policy $\pi(x)$ needs to depend only on the corresponding steering direction $u_x$. It can be shown that for *convex* target sets, the condition of the last theorem turns out to be both sufficient and necessary.

Standard approachability results, as outlined above, require to consider an infinite number of steering directions whenever the reward in non-scalar. The corresponding set of steering policies may turn out to be infinite as well. For the purpose of our learning scheme, we shall require an approaching policy which relies on a *finite* set of steering directions and policies. The following results show that this can indeed be done, possibly requiring to slightly expand the target set. In the following, let $M$ be an upper bound on the magnitude of the expected one-stage reward vector, so that $\|m(s, a, b)\| < M$ for

all $(s, a, b)$ ($|| \cdot ||$ denote the Euclidean norm). We say that a set of vectors $(u_1, \ldots, u_J)$ is an $\epsilon$-cover of the unit ball if for every vector in the unit ball $u$ there exists a vector $u_i$ such that $||u_i - u|| \leq \epsilon$ .

**Theorem 2.2** *Let Assumption 1 hold and suppose that the target set $T \subset \mathbb{R}^k$ satisfies condition (2). Fix $\epsilon > 0$. Let $\{u_1, \ldots, u_J\}$ be an $\epsilon/M$ cover of the unit ball. Suppose that $\pi_i$ is an optimal strategy in the scalar game $\Gamma(u_i)$ $(1 \leq i \leq J)$. Then the following policy approaches $T^\epsilon$, the $\epsilon$-expansion of $T$: If $s_n = s^*$ and $\hat{m}_n \notin T^\epsilon$, then choose $j$ so that $u_{\hat{m}_n}$ is closest to $u_j$ (in Euclidean norm) and play $\pi_j$ until the next visit to state $s^*$; otherwise, play arbitrarily.*

**Proof:** (Outline) The basic observation is that if two directions, $u$ and $u_i$ are close, then $v\Gamma(u)$ and $v\Gamma(u_i)$ are close. Consequently, by playing a strategy which is optimal in $\Gamma(u_i)$ results in a play which is almost optimal in $\Gamma(u)$. Finally we can apply Blackwell's Theorem (2.1) for the expansion of $T$, by noticing that a "good enough" strategy is played in every direction. ∎

**Remark:** It follows immediately from the last theorem that the set $T$ itself (rather than its $\epsilon$-expansion) is approachable with a finite number of steering directions if $T^{-\epsilon}$, the $\epsilon$ shrinkage of $T$, satisfies (2). Equivalently, $T$ is required to satisfy (2) with the 0 on the right-hand-side replaced by $\epsilon > 0$.

# 3    The Multi-Criteria Reinforcement Learning Algorithm

In this section we introduce and prove the convergence of the MCRL (Multi-Criteria Reinforcement Learning) algorithm. We consider the controlled Markov model of Section 2, but here we assume that P1, the learning agent, does not know the model parameters, namely the state transition probabilities and reward functions. A policy of P1 that does not rely on knowledge of these parameters will be referred to as a *learning* policy. P1's task is to approach a given target set $T$, namely to ensure convergence of the average reward vector to this set irrespective of P2's actions.

The proposed learning algorithm relies on the construction of the previous section of approaching policies with a finite number of steering directions. The main idea is to apply a (scalar) learning algorithm for each of the projected games $\Gamma(u_j)$ corresponding to these directions. Recall that each such game is a standard zero-sum stochastic game with average reward. The required learning algorithm for game $\Gamma(u)$ should secure an average reward that is not less than the value $v\Gamma(u)$ of that game.

Consider a zero-sum stochastic game, with reward function $r(s, a, b)$, average reward $\hat{r}_n$ and value v. Assume for simplicity that the initial state is fixed. We say that a learning policy $\pi$ of P1 is $\epsilon$-*optimal* in this game if, for any policy $\sigma$ of P2, the average reward satisfies

$$\liminf_{n \to \infty} \hat{r}_n \geq v - \epsilon \quad P_{\pi\sigma} \text{ a.s.},$$

where $P_{\pi\sigma}$ is the probability measure induced by the algorithm $\pi$, P2's policy $\sigma$ and the game dynamics. Note that P1 may be unable to learn a min-max policy as P2 may play an inferior policy and refrain from playing certain actions, thereby keeping some parts of the game unobserved.

**Remark:** RL for average reward zero-sum stochastic games can be devised in a similar manner to average reward Markov decision processes. For example, a Q-learning based algorithm which combines the ideas of [9] with those of [1] can be devised. An additional assumption that is needed for the analysis is that all actions of both players are used infinitely often. A different type of a scalar algorithm that overcomes this problem is [4]. The algorithm there is similar to the $E^3$ algorithm [8] which is based on explicit exploration-exploitation tradeoff and estimation of the game reward and transition structure.

We now describe the MCRL algorithm that nearly approaches any target set $T$ that satisfies (2). The parameters of the algorithm are $\epsilon$ and $M$. $\epsilon$ is the approximation level and $M$ is a known bound on the norm of the expected reward per step. The goal of the algorithm is to approach $T^\epsilon$, the $\epsilon$ expansion of $T$. There are $J$ learning algorithms that are run in parallel, denoted by $\pi_1, \ldots \pi_J$. The MCRL is described in Figure 3 and is given here as a meta-algorithm (the scalar RL algorithms $\pi_i$ are not specified). When arriving to $s^*$, the decision maker checks if the average reward vector is outside the set $T^\epsilon$. In that case, he switches to an appropriate policy that is intended to "steer" the average reward vector towards the target set. The steering policy $(\pi_j)$ is chosen according to closest

direction $(u_j)$ to the actual direction needed according to the problem geometry. Recall that each $\pi_j$ is actually a learning policy with respect to a scalar reward function. In general, when $\pi_j$ is not played, its learning pauses and the process history during that time is ignored. Note however that some "off-policy" algorithms (such as Q-learning) can learn the optimal policy even while playing a different policy. In that case a more efficient version of the MCRL is suggested, in which learning is performed by all learning policies $\pi_j$ continuously and concurrently.

---

0. Let $u_1, \ldots u_J$ be an $\epsilon/2M$ cover of the unit ball. Initialize $J$ different $\epsilon/2$-optimal scalar algorithms, $\pi_1, \ldots, \pi_J$.
1. If $s_0 \neq s^*$ play arbitrarily until $s_n = s^*$.
2. $(s_n = s^*)$ If $\hat{m}_n \in T^\epsilon$ goto step 1. Else let $i = \arg\min_{1 \leq i \leq J} ||u_i - u_{\hat{m}_n}||_2$.
3. While $s_n \neq s^*$ play according to $\pi_i$, the reward $\pi_i$ receives is $u_i \cdot m_n$.
4. When $s_n = s^*$ goto step 2.

---

Figure 3: The MCRL algorithm

**Theorem 3.1** *Suppose that Assumption 1 holds and the MCRL algorithm is used with $\epsilon$-optimal scalar learning algorithms. If the target set $T$ satisfies (2), then $T^\epsilon$ is approached using MCRL.*

**Proof:** (Outline) If a direction is played infinitely often, then eventually the learned strategy in this direction is nearly optimal. If a direction is not played infinitely often it has a negligible effect on the long term average reward vector. Since the learning algorithms are nearly optimal, then any policy $\pi_j$ that is played infinitely often, eventually attains a (scalar) average reward of $\text{v}\Gamma(u_j) - \epsilon/2$. One can apply Theorem 2.2 for the set $T^{\epsilon/2}$ to verify that the overall policy is an approaching policy for the target set. ∎

Note that for convex target sets the algorithm is consistent in the sense that if the set is approachable then the algorithm attains it.

**Remark:** Multi-criteria Markov Decision Process (MDP) models may be regarded as a special case of the stochastic game model that was considered so far, with P2 eliminated from the problem. The MCRL meta-algorithm of the previous section remains the same for MDPs. Its constituent scalar learning algorithms are now learning algorithms for the optimal polices in average-reward MDPs. These are generally simpler than for the game problem. Examples of optimal or $\epsilon$-optimal algorithms are Q-Learning with persistent exploration [2], Actor-critic schemes [2], an appropriate version of the $E^3$ algorithm [8] and others. In the absence of an adversary, the problem of approaching a set becomes much simpler. Moreover, it can be shown that if a set is approachable then it may be approached using a stationary (possibly randomized) policy. Indeed, any point in feasible set of state-action frequencies may be achieved by such a stationary policy [5]. Thus, alternative learning schemes may be applicable to this problem. Another observation is that all steering policies learned and used within the MCRL may now be *deterministic* stationary policies, which simplifies the implementation of this algorithm.

## 4   Example

Recall the humidity-temperature example from the introduction. Suppose that the system is modelled in such a way that P1 chooses a temperature-humidity curve. Then Nature (modelled as P2) chooses the exact location on the temperature-humidity curve. In Figure 4(a) we show three different temperature-humidity curves, that can be determined by P1 (each defined by a certain strategy of P1 - $f_0, f_1, f_2$). We implemented MCRL algorithm with nine directions. In each direction a version of Littman's Q-learning ([9]), adapted for average cost games, was used. A sample path of the average reward generated by the MCRL algorithm is shown in Figure 4(b). The sample path started at 'S' and finished at 'E'. For this specific run, an even smaller number of directions would have sufficed (up and right). It can be seen that the learning algorithm pushes towards the target set so that the path is mostly on the edge of the target set. Note that in this example a small number of directions was quite enough for approaching the target set.

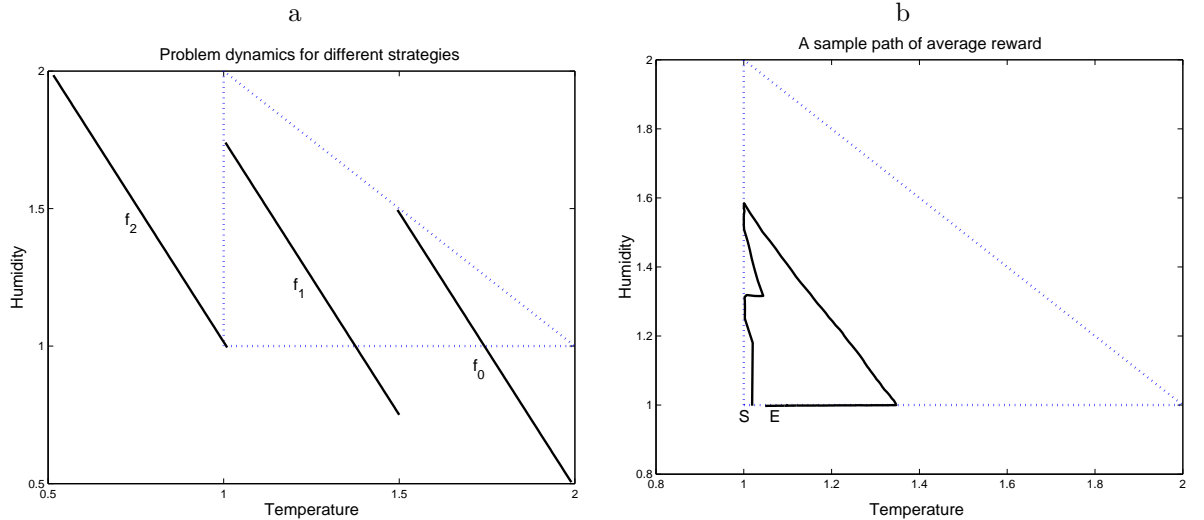

Figure 4: **(a)** Greenhouse problem dynamics. **(b)** A sample path from 'S' to 'E'

## 5    Conclusion

We have presented a learning algorithm that approaches a prescribed target set in multi-dimensional performance space, provided this set satisfies a certain sufficient condition. Our approach essentially relies on the theory of approachability for stochastic games, which is based on the idea of steering the average reward vector towards the target set. We provided a key result stating that a set can be approached to a given precision using only a finite number of steering policies, which may be learned on-line.

An interesting observation regarding the proposed learning algorithm is that the learned optimal polices in each direction are essentially independent of the target set $T$. Thus, the target set need not be fixed in advance and may be modified on-line without requiring a new learning process. This may be especially useful for constrained MDPs.

Of further interest is the question of reduction of the number of steering directions used in the algorithm. In some cases, especially when the requirements embodied by the target set $T$ are not stringent, this number may be quite small compared to the worst-case estimate used above. A possible refinement of the algorithm is to eliminate directions that are not required.

The scaling of he algorithm with the dimension of the reward space is exponential. The problem is that as the dimension increases, exponentially many directions are needed to cover the unit ball. While in general this is necessary, it might happen that considerably less directions are needed. Conditions and algorithms that use much less than exponential number of directions are under current study.

### Acknowledgement

This research was supported by the fund for the promotion of research at the Technion.

## References

[1] J. Abounadi, D. Bertsekas, and V. Borkar. Learning algorithms for markov decision processes with average cost. LIDS-P 2434, Lab. for Info. and Decision Systems, MIT, October 1998.

[2] A.G. Barto and R.S. Sutton. *Reinforcement Learning*. MIT Press, 1998.

[3] D. Blackwell. An analog of the minimax theorem for vector payoffs. *Pacific J. Math.*, 6(1):1–8, 1956.

[4] R.I. Brafman and M. Tennenholtz. A near optimal polynomial time algorithm for learning in certain classes of stochastic games. *Artificial Intelligence*, 121(1-2):31–47, April 2000.

[5] C. Derman. *Finite state Markovian decision processes.* Academic Press, 1970.

[6] J. Filar and K. Vrieze. *Competitive Markov Decision Processes.* Springer Verlag, 1996.

[7] L.P. Kaelbling, M. Littman, and A.W. Moore. Reinforcement learning - a survey. *Journal of Artificial Intelligence Research*, (4):237–285, May 1996.

[8] M. Kearns and S. Singh. Near-optimal reinforcement learning in polynomial time. In *Proc. of the 15th Int. Conf. on Machine Learning*, pages 260–268. Morgan Kaufmann, 1998.

[9] M.L. Littman. Markov games as a framework for multi-agent reinforcement learning. In Morgan Kaufman, editor, *Eleventh International Conference on Machine Learning*, pages 157–163, 1994.

[10] S. Mahadevan. Average reward reinforcement learning: Foundations, algorithms, and empirical results. *Machine Learning*, 22(1):159–196, 1996.

[11] S. Mannor and N. Shimkin. The empirical bayes envelope approach to regret minimization in stochastic games. Technical report EE- 1262, Faculty of Electrical Engineering, Technion, Israel, October 2000.

[12] J.F. Mertens and A. Neyman. Stochastic games. *International Journal of Game Theory*, 10(2):53–66, 1981.

[13] A. Schwartz. A reinforcement learning method for maximizing undiscounted rewards. In *Proceedings of the Tenth International Conference on Machine Learning*, pages 298–305. Morgan Kaufmann, 1993.

[14] N. Shimkin and A. Shwartz. Guaranteed performance regions in markovian systems with competing decision makers. *IEEE Trans. on Automatic Control*, 38(1):84–95, January 1993.
